# PHASE TRANSITIONS IN NEURAL NETWORKS

Joshua Chover
University of Wisconsin, Madison, WI 53706

## ABSTRACT

Various simulations of cortical subnetworks have evidenced something like phase transitions with respect to key parameters. We demonstrate that such transitions must indeed exist in analogous infinite array models. For related finite array models classical phase transitions (which describe steady-state behavior) may not exist, but there can be distinct qualitative changes in ("metastable") transient behavior as key system parameters pass through critical values.

## INTRODUCTION

Suppose that one stimulates a neural network - actual or simulated - and in some manner records the subsequent firing activity of cells. Suppose further that one repeats the experiment for different values of some parameter (p) of the system; and that one finds a "critical value" ($p_c$) of the parameter, such that (say) for values $p > p_c$ the activity tends to be much higher than it is for values $p < p_c$. Then, by analogy with statistical mechanics (where, e.g., p may be temperature, with critical values for boiling and freezing) one can say that the neural network undergoes a "phase transition" at $p_c$. Intracellular phase transitions, parametrized by membrane potential, are well known. Here we consider intercellular phase transitions. These have been evidenced in several detailed cortical simulations: e.g., of the piriform cortex[1] and of the hippocampus[2]. In the piriform case, the parameter p represented the frequency of high amplitude spontaneous EPSPs received by a typical pyramidal cell; in the hippocampal case, the parameter was the ratio of inhibitory to excitatory cells in the system.

By what mechanisms could approach to, and retreat from, a critical value of some parameter be brought about? An intriguing conjecture is that neuromodulators can play such a role in certain networks; temporarily raising or depressing synaptic efficacies[3]. What possible interesting consequences could approach to criticality have for system performance. Good effects could be these: for a network with plasticity, heightened firing response to a stimulus can mean faster changes in synaptic efficacies, which would bring about faster memory storage. More and longer activity could also mean faster access to memory. A bad effect of

near-criticality – depending on other parameters – can be wild, epileptiform activity.

Phase transitions as they might relate to neural networks have been studied by many authors[4]. Here, for clarity, we look at a particular category of network models – abstracted from the piriform cortex setting referred to above – and show the following:

a) For "elementary" reasons, phase transition would have to exist if there were infinitely many cells; and the near-subcritical state involves prolonged cellular firing activity in response to an initial stimulation.

b) Such prolonged firing activity takes place for analogous large finite cellular arrays – as evidenced also by computer simulations.

What we shall be examining is space-time patterns which describe the mid-term transient activity of (Markovian) systems that tend to silence (with high probability) in the long run. (There is no reference to energy functions, nor to long-run stable firing rates – as such rates would be zero in most of our cases.)

In the following models time will proceed in discrete steps. (In the more complicated settings these will be short in comparison to other time constants, so that the effect of quantization becomes smaller.) The parameter p will be the probability that at any given time a given cell will experience a certain amount of excitatory "spontaneous firing" input: by itself this amount will be insufficient to cause the cell to fire, but in conjunction with sufficiently many excitatory inputs from other cells it can assist in reaching firing threshold. (Other related parameters such as average firing threshold value and average efficacy value give similar results.) In all the models there is a refractory period after a cell fires, during which it cannot fire again; and there may be local (shunt type) inhibition by a firing cell on near neighbors as well as on itself – but there is no long-distance inhibition. We look first at limiting cases where there are infinitely many cells and — classically – phase transition appears in a sharp form.

## A "SIMPLE" MODEL

We consider an infinite linear array of similar cells which obey the following rules, pictured in Fig. 1A:

(i) If cell k fires at time n, then it must be silent at time n+1;

(ii) if cell k is silent at time n but both of its neighbors k-1 and k+1 do fire at time n, then cell k fires at time n+1;

(iii) if cell k is silent at time n and just one of its neighbors (k-1 or k+1) fires at time n, then cell k will fire at time n+1 with probability p and not fire with probability 1-p, independently of similar decisions at other cells and at other times.

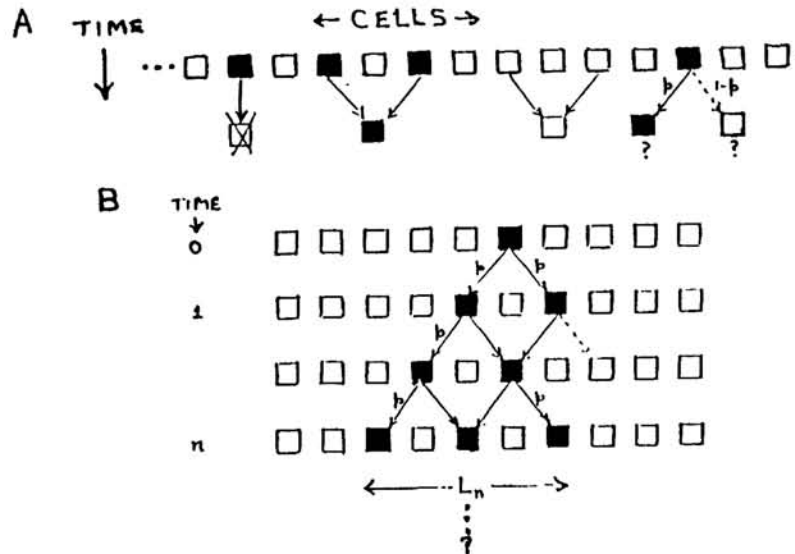

Fig. 1. "Simple model". A: firing rules; cells are represented
horizontally, time proceeds downwards; filled squares
denote firing. B: sample development.

Thus, effectively, signal propagation speed here is one cell
per unit time, and a cell's firing threshold value is 2 (EPSP
units). If we stimulate one cell to fire at time $n=0$, will its
influence necessarily die out or can it go on forever? (See
Fig. 1B.) For an answer we note that in this simple case the
firing pattern (if any) at time $n$ must be an alternating stretch
of firing/silent cells of some length, call it $L_n$. Moreover,

$L_{n+1} = L_n + 2$ with probability $p^2$ (when there are sponteneous
firing assists on both ends of the stretch), or $L_{n+1} = L_n - 2$ with

probability $(1-p)^2$ (when there is no assist at either end of the
stretch), or $L_{n+1} = L_n$ with probability $2p(1-p)$ (when there is
an assist at just one end of the stretch).

Starting with any finite alternating stretch $L_0$, the
successive values $L_n$ constitute a "random walk" among the

nonnegative integers. Intuition and simple analysis[5] lead to the
same conclusion: if the probability for $L_n$ to decrease $((1-p)^2)$

is greater than that for it to increase $(p^2)$ – i.e. if the average
step taken by the random walk is negative – then ultimately $L_n$

will reach 0 and the firing response dies out. Contrariwise, if

$p^2 > (1-p)^2$ then the $L_n$ can drift to even higher values with positive probability. In Fig. 2A we sketch the probability for ultimate die-out as a function of p; and in Fig. 2B, the average time until die out. Figs. 2A and B show a classic example of phase transition $(p_c = 1/2)$ for this infinite array.

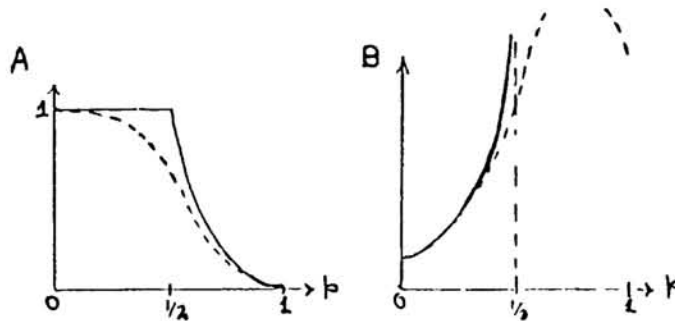

Fig. 2. Critical behavior. A: probability of ultimate die out (or of reaching other traps, in finite array case).
B: average time until die-out (or for reaching other traps). Solid curves refer to an infinite array; dashed, to finite arrays.

## MORE COMPLEX MODELS

For an infinite linear array of cells, as sketched in Fig. 3 , we describe now a much more general (and hopefully more realistic) set of rules:

(i') A cell cannot fire, nor receive excitatory inputs, at time n if it has fired at any time during the preceding $m_R$ time units (refraction and feedback inhibition).

(ii') Each cell x has a local "inhibitory neighborhood" consisting of a number (j) of cells to its immediate right and left. The given cell x cannot fire or receive excitatory inputs at time n if any other cell y in its inhibitory neighborhood has fired at any time between t and $t+m_I$ units preceding n, where t is the time it would take for a message to travel from y to x at a speed of $v_I$ cells per unit time. (This rule represents local shunt-type inhibition.)

(iii') Each cell x has an "excitatory neighborhood" consisting of a number (e) of cells to the immediate right and left of its inhibitory neighborhood. If a cell y in that neighborhood fires at a certain time, that firing causes a unit impulse to travel to cell x at a speed of $v_E$ cells per unit time. The impulse is received at x subject to rules (i') and (ii').

(iv') All cells share a "firing threshold" value θ and an "integration time constant." s (s < θ). In addition each cell, at each time n and independently of other times and other cells, can receive a random amount $X_n$ of "spontaneous excitatory input".

The variable $X_n$ can have a general distribution; however, for simplicity we suppose here that it assumes only one of two values: b or 0, with probabilities p and 1-p respectively. (We suppose that b < θ, so that the spontaneous "assist" itself is insufficient for firing.) The above quantities enter into the following firing rule: a cell will fire at time n if it is not prevented by rules (i') and (ii') and if the total number of inputs from other cells, received during the integration "window" lasting between times n-s+1 and n inclusive, plus the assist $X_n$, equals or exceeds the threshold θ.

(The propagation speeds $v_I$ and $V_E$ and the neighborhoods are here given left-right symmetry merely for ease in exposition.)

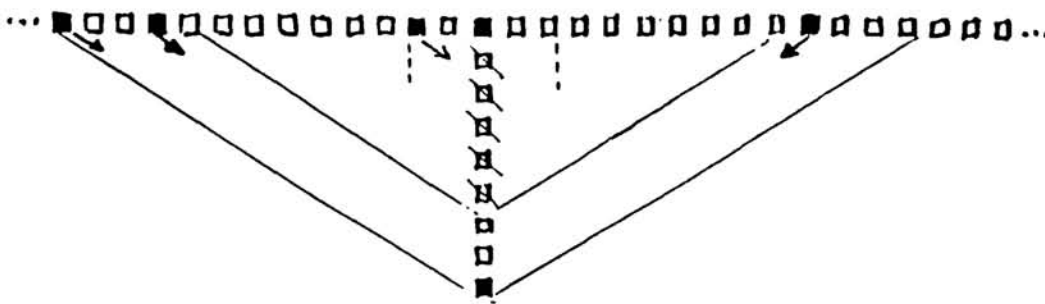

Fig. 3. Message travel in complex model:  see text rules (i')-(iv').

Will such a model display phase transition at some critical value of the spontaneous firing frequency p ? The dependence of responses upon the initial conditions and upon the various parameters is intricate and will affect the answer. We briefly discuss here conditions under which the answer is again yes.

(1) For a given configuration of parameters and a given initial stimulation (of a stretch of contiguous cells) we compare the development of the model's firing response first to that of an auxiliary "more active" system: Suppose that $L_n$ now denotes the distance at time n between the left- and right-most cells which are either firing or in refractory mode. Because no cell can fire without influence from others and because such influence travels at a given speed, there is a maximal amount (D) whereby $L_{n+1}$ can exceed $L_n$. There is also a maximum probability Q(p) - which

depends on the spontaneous firing parameter p – that $L_{n+1} \geq L_n$ (whatever n). We can compare $L_n$ with a random walk "$A_n$" defined so that $A_{n+1} = A_n+D$ with probability $Q(p)$ and $A_{n+1} = A_n-1$ with probability $1-Q(p)$. At each transition, $A_n$ is more likely to increase than $L_n$. Hence $L_n$ is more likely to die out than $A_n$. In the many cases where $Q(p)$ tends to zero as p does, the average step size of $A_n$ (viz., $DQ(p)+(-1)(1-Q(b))$) will become negative for p below a "critical" value $p_a$. Thus, as in the "simple" model above, the probability of ultimate die-out for the $A_n$, hence also for the $L_n$ of the complex model, will be 1 when $0 \leq p < p_a$.

(2) There will be a phase transition for the complex model if its probability of die out – given the same parameters and initial stimulation is in (1) – becomes less than 1 for some p values with $p_a < p < 1$. Comparison of the complex process with a simpler "less active" process is difficult in general. However, there are parameter configurations which ultimately can channel all or part of the firing activity into a (space-time) sublattice analogous to that in Fig. 1. Fig. 4 illustrates such a case. For p sufficiently large there is positive probability that the activity will not die out, just as in the "simple" model.

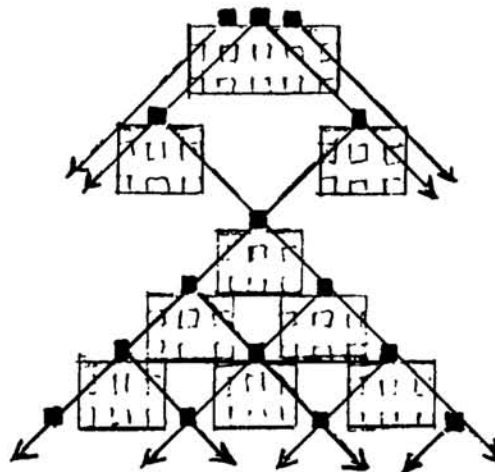

Fig. 4. Activity on a sublattice. (Parameter values: j=2, e=6, $M_R$=2, $M_I$=1, $V_R=V_I$=1, θ=3, s=2, and b=1.) Rectangular areas indicate refraction/inhibition; diagonal lines, excitatory influence.

## LARGE FINITE ARRAYS

Consider now a large finite array of  N  cells, again as
sketched in Fig. 3 ; and operating according to rules similar to
(i')-(iv') above, with suitable modifications near the edges.
Appropriately encoded, its activity can be described by a (huge)
Markov transition matrix, and - depending on the initial
stimulation - must tend[5] to one of a set of steady-state
distributions over firing patterns. For example, ($\alpha$) if  N  is
odd and the rules are those for Fig. 1, then extinction is the
unique steady state, for any  p < 1  (since the  $L_n$  form a random
walk with "reflecting" upper barrier). But, ($\beta$) if  N  is even
and the cells are arranged in a ring, then, for any  p  with
0 < p < 1,  both extinction and an alternate flip-flop firing
pattern of period 2 are "traps" for the system - with relative long
run probabilities determined by the initial state. See the dashed
line in Fig. 2A for the extinction probability in the  ($\beta$)  case,
and in Fig. 2B for the expected time until hitting a trap in the
($\alpha$)  case  ($p < \frac{1}{2}$)  and the  ($\beta$)  case.

What qualitative properties related to phase transition and
critical  p  values carry over from the infinite to the finite
array case? The  ($\alpha$)  example above shows that long term activity
may now be the same for all  0 < p < 1  but that parameter
intervals can exist whose key feature is a particularly large
expected time before the system hits a trap. (Again, the critical
region can depend upon the initial stimulation.) Prior to being
trapped the system spends its time among many states in a kind of
"metastable" equilibrium. (We have some preliminary theoretical
results on this conditional equilibrium and on its relation to the
infinite array case. See also Ref. 6 concerning time scales for
which certain corresponding infinite and finite stochastic automata
systems display similar behavior.)
Simulation of models satisfying rules (i')-(iv') does indeed
display large changes in length of firing activity corresponding to
parameter changes near a critical value. See Fig. 5 for a typical
example: As a function of  p,  the expected time until the system
is trapped (for the given parameters) rises approximately linearly
in the interval  .05<p<.12,  with most runs resulting in extinction
- as is the case in Fig. 5A at time  n=115  (for  p=.10). But for
p>.15  a relatively rigid patterning sets in which leads with high
probability to very long runs or to traps other than extinction -
as is the case in Fig. 5B  (p=.20)  where the run is arbitrarily
truncated at  n=525. (The patterning is highly influenced by the
large size of the excitatory neighborhoods.)

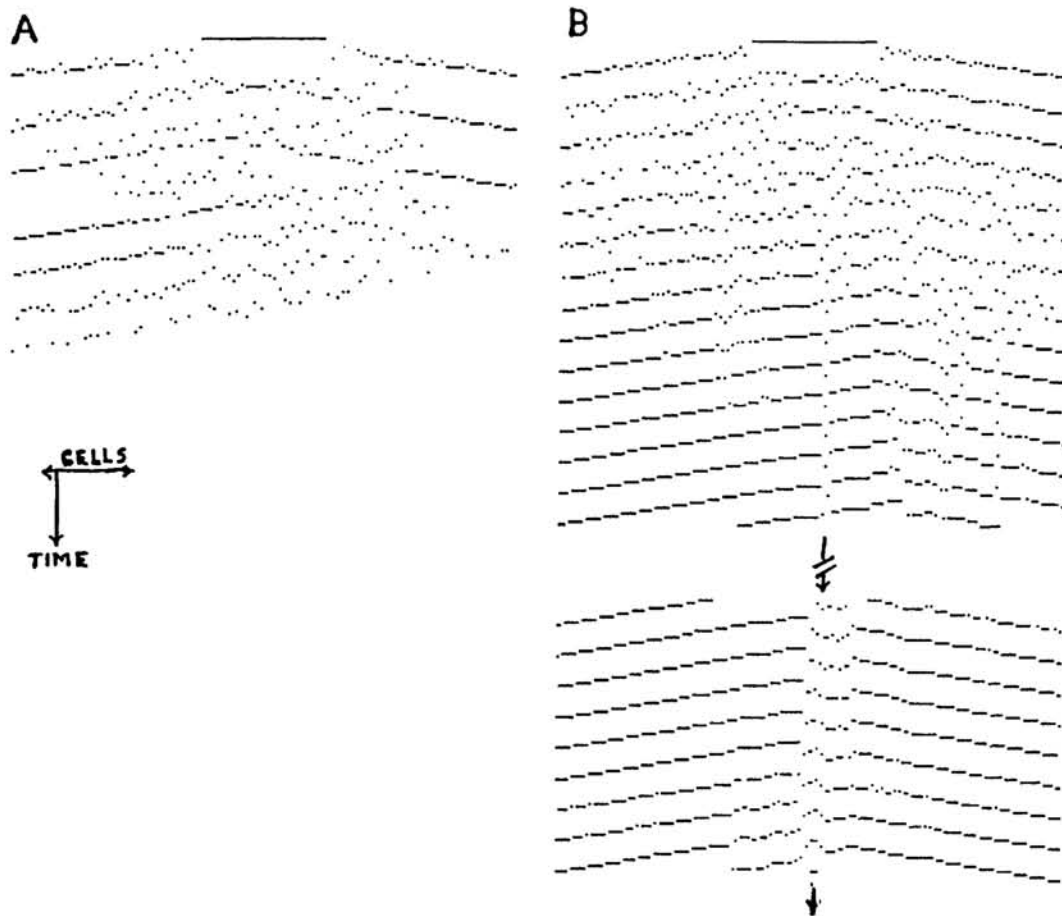

Fig. 5. Space time firing patterns for one configuration of basic
parameters. (There are 200 cells; j=2, e=178, $M_R$=10,
$M_I$=9, $V_R$=$V_I$=7, $\theta$=25, s=2, and b=12; 50 are stimulated
initially.) A: p=.10. B: p=.20.

CONCLUSION

Mechanisms such as neuromodulators, which can (temporarily)
bring spontaneous firing levels – or synaptic efficacies, or
average firing thresholds, or other similar parameters – to
near-critical values, can thereby induce large amplification of
response activity to selected stimuli. The repertoire of such
responses is an important aspect of the system's function.

[Acknowledgement: Thanks to C. Bezuidenhout and J. Kane for help with simulations.]

## REFERENCES

1. M. Wilson, J. Bower, J. Chover, L. Haberly, 16th Neurosci. Soc. Mtg. Abstr. 370.11 (1986).
2. R. D. Traub, R. Miles, R.K.S. Wong, 16th Neurosci. Soc. Mtg. Abstr. 196.12 (1986).
3. A. Selverston, this conference, also, Model Neural Networks and Behavior, Plenum (1985); E. Marder, S. Hooper, J. Eisen, Synaptic Function, Wiley (1987) p.305.
4. E.g.: W. Kinzel, Z. Phys. B58, p. 231 (1985); A. Noest. Phys. Rev. Let. 57(1), p. 90 (1986); R. Durrett (to appear); G. Carpenter, J. Diff. Eqns. 23, p.335 (1977); G. Ermentraut, S. Cohen, Biol. Cyb. 34, p.137 (1979); H. Wilson, S. Cowan, Biophys. J. 12 (1972).
5. W. Feller, An Introd. to Prob. Th'y. and Appl'ns. I. Wiley (1968) Ch. 14, 15.
6. T. Cox and A. Graven (to appear).
